# Natural Language Grammar Induction using a Constituent-Context Model

**Dan Klein** and **Christopher D. Manning**
Computer Science Department
Stanford University
Stanford, CA 94305-9040
{*klein, manning*}*@cs.stanford.edu*

## Abstract

This paper presents a novel approach to the unsupervised learning of syntactic analyses of natural language text. Most previous work has focused on maximizing likelihood according to generative PCFG models. In contrast, we employ a *simpler* probabilistic model over trees based directly on constituent identity and linear context, and use an EM-like iterative procedure to induce structure. This method produces much higher quality analyses, giving the best published results on the ATIS dataset.

## 1 Overview

To enable a wide range of subsequent tasks, human language sentences are standardly given tree-structure analyses, wherein the nodes in a tree dominate contiguous spans of words called *constituents*, as in figure 1(a). Constituents are the linguistically coherent units in the sentence, and are usually labeled with a constituent category, such as noun phrase (NP) or verb phrase (VP). An aim of grammar induction systems is to figure out, given just the sentences in a corpus $S$, what tree structures correspond to them. In this sense, the grammar induction problem is an incomplete data problem, where the complete data is the corpus of trees $T$, but we only observe their yields $S$. This paper presents a new approach to this problem, which gains leverage by directly making use of constituent contexts.

It is an open problem whether entirely unsupervised methods can produce linguistically accurate parses of sentences. Due to the difficulty of this task, the vast majority of statistical parsing work has focused on supervised learning approaches to parsing, where one uses a treebank of fully parsed sentences to induce a model which parses unseen sentences [7, 3]. But there are compelling motivations for unsupervised grammar induction. Building supervised training data requires considerable resources, including time and linguistic expertise. Investigating unsupervised methods can shed light on linguistic phenomena which are implicit within a supervised parser's supervisory information (e.g., unsupervised systems often have difficulty correctly attaching subjects to verbs above objects, whereas for a supervised parser, this ordering is implicit in the supervisory information). Finally, while the presented system makes no claims to modeling human language acquisition, results on whether there is enough information in sentences to recover their structure are important data for linguistic theory, where it has standardly been assumed that the information in the data is deficient, and strong innate knowledge is required for language acquisition [4].

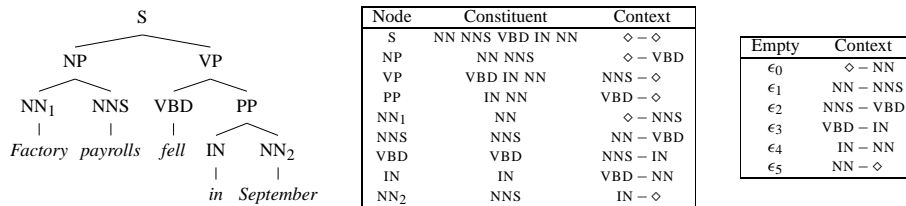

Figure 1: Example parse tree with the constituents and contexts for each tree node.

## 2 Previous Approaches

One aspect of grammar induction where there has already been substantial success is the induction of parts-of-speech. Several different distributional clustering approaches have resulted in relatively high-quality clusterings, though the clusters' resemblance to classical parts-of-speech varies substantially [9, 15]. For the present work, we take the part-of-speech induction problem as solved and work with sequences of parts-of-speech rather than words. In some ways this makes the problem easier, such as by reducing sparsity, but in other ways it complicates the task (even supervised parsers perform relatively poorly with the actual words replaced by parts-of-speech).

Work attempting to induce tree structures has met with much less success. Most grammar induction work assumes that trees are generated by a symbolic or probabilistic context-free grammar (CFG or PCFG). These systems generally boil down to one of two types. Some fix the structure of the grammar in advance [12], often with an aim to incorporate linguistic constraints [2] or prior knowledge [13]. These systems typically then attempt to find the grammar production parameters $\Theta$ which maximize the likelihood $P(S|\Theta)$ using the inside-outside algorithm [1], which is an efficient (dynamic programming) instance of the EM algorithm [8] for PCFGs. Other systems (which have generally been more successful) incorporate a structural search as well, typically using a heuristic to propose candidate grammar modifications which minimize the joint encoding of data and grammar using an MDL criterion, which asserts that a good analysis is a short one, in that the joint encoding of the grammar and the data is compact [6, 16, 18, 17]. These approaches can also be seen as likelihood maximization where the objective function is the *a posteriori* likelihood of the grammar given the data, and the description length provides a structural prior.

The "compact grammar" aspect of MDL is close to some traditional linguistic argumentation which at times has argued for minimal grammars on grounds of analytical [10] or cognitive [5] economy. However, the primary weakness of MDL-based systems does not have to do with the objective function, but the search procedures they employ. Such systems end up growing structures greedily, in a bottom-up fashion. Therefore, their induction quality is determined by how well they are able to heuristically predict what local intermediate structures will fit into good final global solutions.

A potential advantage of systems which fix the grammar and only perform parameter search is that they do compare complete grammars against each other, and are therefore able to detect which give rise to systematically compatible parses. However, although early work showed that small, artificial CFGs could be induced with the EM algorithm [12], studies with large natural language grammars have generally suggested that completely unsupervised EM over PCFGs is ineffective for grammar acquisition. For instance, Carroll and Charniak [2] describe experiments running the EM algorithm from random starting points, which produced widely varying learned grammars, almost all of extremely poor quality.[1]

It is well-known that EM is only locally optimal, and one might think that the locality of the search procedure, not the objective function, is to blame. The truth is somewhere in between. There are linguistic reasons to distrust an ML objective function. It encourages the symbols and rules to align in ways which maximize the truth of the conditional independence assumptions embodied by the PCFG. The symbols and rules of a natural language grammar, on the other hand, represent syntactically and semantically coherent units, for which a host of linguistic arguments have been made [14]. None of these have anything to do with conditional independence; traditional linguistic constituency reflects only grammatical regularities and possibilities for expansion. There are expected to be strong connections across phrases (such as dependencies between verbs and their selected arguments). It could be that ML over PCFGs and linguistic criteria align, but in practice they do not always seem to. Experiments with both artificial [12] and real [13] data have shown that starting from fixed, correct (or at least linguistically reasonable) structure, EM produces a grammar which has higher log-likelihood than the linguistically determined grammar, but lower parsing accuracy.

However, we additionally conjecture that EM over PCFGs fails to propagate contextual cues efficiently. The reason we expect an algorithm to converge on a good PCFG is that there seem to be coherent categories, like noun phrases, which occur in distinctive environments, like between the beginning of the sentence and the verb phrase. In the inside-outside algorithm, the product of inside and outside probabilities $\alpha_j(p,q)\beta_j(p,q)$ is the probability of generating the sentence with a $j$ constituent spanning words $p$ through $q$: the outside probability captures the environment, and the inside probability the coherent category. If we had a good idea of what VPs and NPs looked like, then if a novel NP appeared in an NP context, the outside probabilities should pressure the sequence to be parsed as an NP. However, what happens early in the EM procedure, when we have no real idea about the grammar parameters? With randomly-weighted, complete grammars over a symbol set $X$, we have observed that a frequent, short, noun phrase sequence often does get assigned to some category $x$ early on. However, since there is not a clear overall structure learned, there is only very weak pressure for other NPs, even if they occur in the same positions, to also be assigned to $x$, and the reestimation process goes astray. To enable this kind of constituent-context pressure to be effective, we propose the model in the following section.

## 3 The Constituent-Context Model

We propose an alternate parametric family of models over trees which is better suited for grammar induction. Broadly speaking, inducing trees like the one shown in figure 1(a) can be broken into two tasks. One is deciding constituent identity: where the brackets should be placed. The second is deciding what to label the constituents. These tasks are certainly correlated and are usually solved jointly. However, the task of labeling chosen brackets is essentially the same as the part-of-speech induction problem, and the solutions cited above can be adapted to cluster constituents [6]. The task of deciding brackets, is the harder task. For example, the sequence DT NN IN DT NN ([*the man in the moon*]) is virtually always a noun phrase when it is a constituent, but it is only a constituent 66% of the time, because the IN DT NN is often attached elsewhere ([*we* [*sent a man*] [*to the moon*]]). Figure 2(a)

probabilities. Figure 4 shows that the resulting grammar (DEP-PCFG) is not as bad as conventional wisdom suggests. Carroll and Charniak are right to observe that the search spaces is riddled with pronounced local maxima, and EM does not do nearly so well when randomly initialized. The *need* for random seeding in using EM over PCFGs is two-fold. For some grammars, such as one over a set $X$ of non-terminals in which any $x_1 \rightarrow x_2\, x_3$, $x_i \in X$ is possible, it is needed to break symmetry. This is not the case for dependency grammars, where symmetry is broken by the yields (e.g., a sentence *noun verb* can only be covered by a noun or verb projection). The second reason is to start the search from a random region of the space. But unless one does many random restarts, the uniform starting condition is better than most extreme points in the space, and produces superior results.

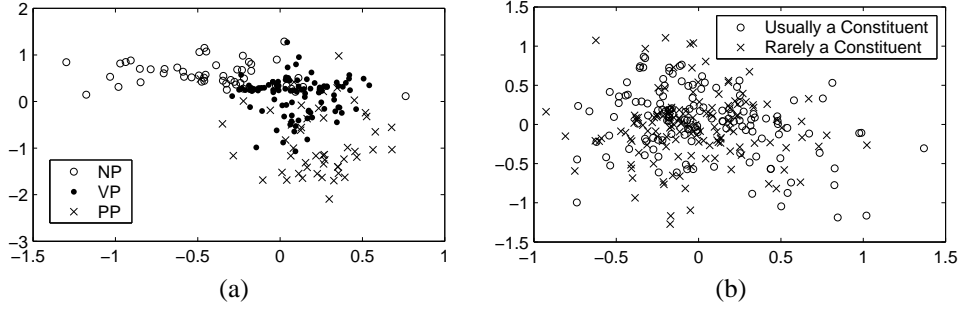

Figure 2: The most frequent examples of (a) different constituent labels and (b) constituents and non-constituents, in the vector space of linear contexts, projected onto the first two principal components. Clustering is effective for labeling, but not detecting constituents.

shows the 50 most frequent constituent sequences of three types, represented as points in the vector space of their contexts (see below), projected onto their first two principal components. The three clusters are relatively coherent, and it is not difficult to believe that a clustering algorithm could detect them in the unprojected space. Figure 2(a), however, shows 150 sequences which are parsed as constituents at least 50% of the time along with 150 which are not, again projected onto the first two components. This plot at least suggests that the constituent/non-constituent classification is less amenable to direct clustering.

Thus, it is important that an induction system be able to detect constituents, either implicitly or explicitly. A variety of methods of constituent detection have been proposed [11, 6], usually based on information-theoretic properties of a sequence's distributional context. However, here we rely entirely on the following two simple assumptions: (i) constituents of a parse do not cross each other, and (ii) constituents occur in constituent contexts. The first property is self-evident from the nature of the parse trees. The second is an extremely weakened version of classic linguistic constituency tests [14].

Let $\sigma$ be a terminal sequence. Every occurrence of $\sigma$ will be in some linear context $c(\sigma) = x \sigma y$, where $x$ and $y$ are the adjacent terminals or sentence boundaries. Then we can view any tree $t$ over a sentence $s$ as a collection of sequences and contexts, one of each for every node in the tree, plus one for each inter-terminal empty span, as in figure 1(b). Good trees will include nodes whose yields frequently occur as constituents and whose contexts frequently surround constituents. Formally, we use a conditional exponential model of the form:

$$P(t|s, \Theta) = \frac{\exp(\sum_{(\sigma,c)\in t} \lambda_\sigma f_\sigma + \lambda_c f_c)}{\sum_{t:\text{yield}(t)=s} \exp(\sum_{(\sigma,c)\in t} \lambda_\sigma f_\sigma + \lambda_c f_c)}$$

We have one feature $f_\sigma(t)$ for each sequence $\sigma$ whose value on a tree $t$ is the number of nodes in $t$ with yield $\sigma$, and one feature $f_c(t)$ for each context $c$ representing the number of times $c$ is the context of the yield of some node in the tree.[2] No joint features over $c$ and $\sigma$ are used, and, unlike many other systems, there is no distinction between constituent types.

We model only the conditional likelihood of the trees, $P(T|S, \Theta)$, where $\Theta = \{\lambda_\sigma, \lambda_c\}$. We then use an iterative EM-style procedure to find a local maximum $P(T|S, \Theta)$ of the completed data (trees) $T$ $(P(T|S, \Theta) = \prod_{t\in T, s=\text{yield}(t)} P(t|s, \Theta))$. We initialize $\Theta$ such that each $\lambda$ is zero and initialize $T$ to any arbitrary set of trees. In alternating steps, we first fix the parameters $\Theta$ and find the most probable single tree structure $t^*$ for each sentence $s$ according to $P(t|s, \Theta)$, using a simple dynamic program. For any $\Theta$ this produces the

set of parses $T^*$ which maximizes $P(T|S, \Theta)$. Since $T^*$ maximizes this quantity, if $T'$ is the former set of trees, $P(T^*|S, \Theta) \geq P(T'|S, \Theta)$. Second, we fix the trees and estimate new parameters $\Theta$. The task of finding the parameters $\Theta^*$ which maximize $P(T|S, \Theta)$ is simply the well-studied task of fitting our exponential model to maximize the conditional likelihood of the fixed parses. Running, for example, a conjugate gradient (CG) ascent on $\Theta$ will produce the desired $\Theta^*$. If $\Theta'$ is the former parameters, then we will have $P(T|S, \Theta^*) \geq P(T|S, \Theta')$. Therefore, each iteration will increase $P(T|S, \Theta)$ until convergence.[3] Note that our parsing model is not a generative model, and this procedure, though clearly related, is not exactly an instance of the EM algorithm. We merely guarantee that the conditional likelihood of the data *completions* is increasing. Furthermore, unlike in EM where each iteration increases the marginal likelihood of the fixed observed data, our procedure increases the conditional likelihood of a changing complete data set, with the completions changing at every iteration as we reparse.

Several implementation details were important in making the system work well. First, tie-breaking was needed, most of all for the first round. Initially, the parameters are zero, and all parses are therefore equally likely. To prevent bias, all ties were broken randomly.

Second, like so many statistical NLP tasks, smoothing was vital. There are features in our model for arbitrarily long yields and most yield types occurred only a few times. The most severe consequence of this sparsity was that initial parsing choices could easily become frozen. If a $\lambda_\sigma$ for some yield $\sigma$ was either $\gg 0$ or $\ll 0$, which was usually the case for rare yields, $\sigma$ would either be locked into always occurring or never occurring, respectively. Not only did we want to push the $\lambda_\sigma$ values close to zero, we also wanted to account for the fact that most spans are *not* constituents.[4] Therefore, we expect the distribution of the $\lambda_\sigma$ to be skewed towards low values.[5] A greater amount of smoothing was needed for the first few iterations, while much less was required in later iterations.

Finally, parameter estimation using a CG method was slow and difficult to smooth in the desired manner, and so we used the smoothed relative frequency estimates $\lambda_\sigma = \text{count}(f_\sigma)/(\text{count}(\sigma) + M)$ and $\lambda_c = \text{count}(f_c)/(\text{count}(c) + N)$. These estimates ensured that the $\lambda$ values were between 0 and 1, and gave the desired bias towards non-constituency. These estimates were fast and surprisingly effective, but do not guarantee non-decreasing conditional likelihood (though the conditional likelihood was increasing in practice).[6]

## 4   Results

In all experiments, we used hand-parsed sentences from the Penn Treebank. For training, we took the approximately 7500 sentences in the Wall Street Journal (WSJ) section which contained 10 words or fewer after the removal of punctuation. For testing, we evaluated the system by comparing the system's parses for those same sentences against the supervised parses in the treebank. We consider each parse as a *set* of constituent brackets, discarding all trivial brackets.[7] We calculated the precision and recall of these brackets against the treebank parses in the obvious way.

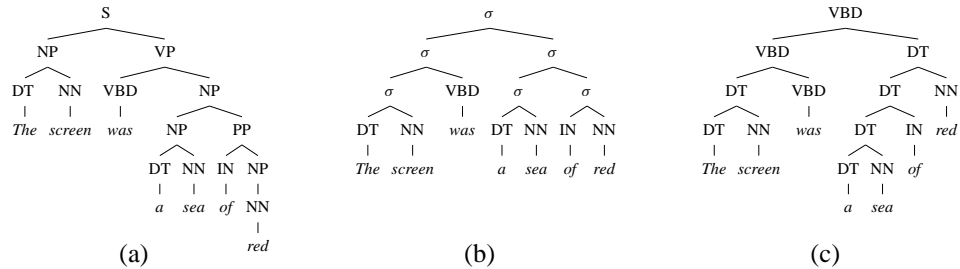

Figure 3: Alternate parse trees for a sentence: (a) the Penn Treebank tree (deemed correct), (b) the one found by our system CCM, and (c) the one found by DEP-PCFG.

| Method | UP | UR | F$_1$ | NP UR | PP UR | VP UR |
|---|---|---|---|---|---|---|
| LBRANCH | 20.5 | 24.2 | 22.2 | 28.9 | 6.3 | 0.6 |
| RANDOM | 29.0 | 31.0 | 30.0 | 42.8 | 23.6 | 26.3 |
| DEP-PCFG | 39.5 | 42.3 | 40.9 | 69.7 | 44.1 | 22.8 |
| RBRANCH | 54.1 | 67.5 | 60.0 | 38.3 | 44.5 | **85.8** |
| CCM | **60.1** | **75.4** | **66.9** | **83.8** | **71.6** | 66.3 |
| UBOUND | 78.2 | 100.0 | 87.8 | 100.0 | 100.0 | 100.0 |

(a)

| System | UP | UR | F$_1$ | CB |
|---|---|---|---|---|
| EMILE | 51.6 | 16.8 | 25.4 | **0.84** |
| ABL | 43.6 | 35.6 | 39.2 | 2.12 |
| CDC-40 | 53.4 | 34.6 | 42.0 | 1.46 |
| RBRANCH | 39.9 | 46.4 | 42.9 | 2.18 |
| CCM | **54.4** | **46.8** | **50.3** | 1.61 |

(b)

Figure 4: Comparative accuracy on WSJ sentences (a) and on the ATIS corpus (b). UR = unlabeled recall; UP = unlabeled precision; F$_1$ = the harmonic mean of UR and UP; CB = crossing brackets. Separate recall values are shown for three major categories.

To situate the results of our system, figure 4(a) gives the values of several parsing strategies. CCM is our constituent-context model. DEP-PCFG is a dependency PCFG model [2] trained using the inside-outside algorithm. Figure 3 shows sample parses to give a feel for the parses the systems produce. We also tested several baselines. RANDOM parses randomly. This is an appropriate baseline for an unsupervised system. RBRANCH always chooses the right-branching chain, while LBRANCH always chooses the left-branching chain. RBRANCH is often used as a baseline for supervised systems, but exploits a systematic right-branching tendency of English. An unsupervised system has no *a priori* reason to prefer right chains to left chains, and LBRANCH is well worse than RANDOM. A system need not beat RBRANCH to claim partial success at grammar induction. Finally, we include an upper bound. All of the parsing strategies and systems mentioned here give fully binary-branching structures. Treebank trees, however, need not be fully binary-branching, and generally are not. As a result, there is an upper bound UBOUND on the precision and F$_1$ scores achievable when structurally confined to binary trees.

Clearly, CCM is parsing much better than the RANDOM baseline and the DEP-PCFG induced grammar. Significantly, it also out-performs RBRANCH in both precision and recall, and, to our knowledge, it is the first unsupervised system to do so. To facilitate comparison with other recent systems, figure 4(b) gives results where we trained as before but used (all) the sentences from the distributionally different ATIS section of the treebank as a test set. For this experiment, precision and recall were calculated using the EVALB system of measuring precision and recall (as in [6, 17]) – EVALB is a standard for parser evaluation, but complex, and unsuited to evaluating unlabeled constituency. EMILE and ABL are lexical systems described in [17]. The results for CDC-40, from [6], reflect training on much more data (12M words). Our system is superior in terms of both precision and recall (and so F$_1$).

These figures are certainly not all that there is to say about an induced grammar; there are a number of issues in how to interpret the results of an unsupervised system when comparing with treebank parses. Errors come in several kinds. First are innocent sins of commission. Treebank trees are very flat; for example, there is no analysis of the inside of many short noun phrases ([*two hard drives*] rather than [*two* [*hard drives*]]). Our system gives a

| Sequence | Example | CORRECT | FREQUENCY | ENTROPY | DEP-PCFG | CCM |
|---|---|---|---|---|---|---|
| DT NN | the man | 1 | 2 | 2 | 1 | 1 |
| NNP NNP | United States | 2 | 1 | – | 2 | 2 |
| CD CD | 4 1/2 | 3 | 9 | – | 5 | 5 |
| JJ NNS | daily yields | 4 | 7 | 3 | 4 | 4 |
| DT JJ NN | the top rank | 5 | – | – | 7 | 6 |
| DT NNS | the people | 6 | – | – | – | 10 |
| JJ NN | plastic furniture | 7 | 3 | 7 | 3 | 3 |
| CD NN | 12 percent | 8 | – | – | – | 9 |
| IN NN | on Monday | 9 | – | 9 | – | – |
| IN DT NN | for the moment | 10 | – | – | – | – |
| NN NNS | fire trucks | 11 | – | 6 | – | 8 |
| NN NN | fire truck | 22 | 8 | 10 | – | 7 |
| TO VB | to go | 26 | – | 1 | 6 | – |
| DT JJ | ?the big | 78 | 6 | – | – | – |
| IN DT | *of the | 90 | 4 | – | 10 | – |
| PRP VBZ | ?he says | 95 | – | – | 8 | – |
| PRP VBP | ?they say | 180 | – | – | 9 | – |
| NNS VBP | ?people are | =350 | – | 4 | – | – |
| NN VBZ | ?value is | =532 | 10 | 5 | – | – |
| NN IN | *man from | =648 | 5 | – | – | – |
| NNS VBD | ?people were | =648 | – | 8 | – | – |

Figure 5: Top non-trivial sequences by actual treebank constituent counts, linear frequency, scaled context entropy, and in DEP-PCFG and CCM learned models' parses.

(usually correct) analysis of the insides of such NPs, for which it is penalized on precision (though not recall or crossing brackets). Second are systematic alternate analyses. Our system tends to form modal verb groups and often attaches verbs first to pronoun subjects rather than to objects. As a result, many VPs are systematically incorrect, boosting crossing bracket scores and impacting VP recall. Finally, the treebank's grammar is sometimes an arbitrary, and even inconsistent standard for an unsupervised learner: alternate analyses may be just as good.[8] Notwithstanding this, we believe that the treebank parses have enough truth in them that parsing scores are a useful component of evaluation.

Ideally, we would like to inspect the quality of the grammar directly. Unfortunately, the grammar acquired by our system is implicit in the learned feature weights. These are not by themselves particularly interpretable, and not directly comparable to the grammars produced by other systems, except through their functional behavior. Any grammar which parses a corpus will have a distribution over which sequences tend to be analyzed as constituents. These distributions can give a good sense of what structures are and are not being learned. Therefore, to supplement the parsing scores above, we examine these distributions.

Figure 5 shows the top scoring constituents by several orderings. These lists do not say very much about how long, complex, recursive constructions are being analyzed by a given system, but grammar induction systems are still at the level where major mistakes manifest themselves in short, frequent sequences. CORRECT ranks sequences by how often they occur *as constituents* in the treebank parses. DEP-PCFG and CCM are the same, but use counts from the DEP-PCFG and CCM parses. As a baseline, FREQUENCY lists sequences by how often they occur anywhere in the sentence yields. Note that the sequence IN DT (e.g., "of the") is high on this list, and is a typical error of many early systems. Finally, ENTROPY is the heuristic proposed in [11] which ranks by context entropy. It is better in practice than FREQUENCY, but that isn't self-evident from this list. Clearly, the lists produced by the CCM system are closer to correct than the others. They look much like a censored version of the FREQUENCY list, where sequences which do not co-exist with higher-ranked ones have been removed (e.g., IN DT often crosses DT NN). This observation may explain a good part of the success of this method.

Another explanation for the surprising success of the system is that it exploits a deep fact about language. Most long constituents have some short, frequent equivalent, or *proform*, which occurs in similar contexts [14]. In the very common case where the proform is a single word, it is guaranteed constituency, which will be transmitted to longer sequences

via shared contexts (categories like PP which have infrequent proforms are not learned well unless the empty sequence is in the model – interestingly, the empty sequence appears to act as the proform for PPs, possibly due to the highly optional nature of many PPs).

# 5   Conclusions

We have presented an alternate probability model over trees which is based on simple assumptions about the nature of natural language structure. It is driven by the explicit transfer between sequences and their contexts, and exploits both the proform phenomenon and the fact that good constituents must tile in ways that systematically cover the corpus sentences without crossing. The model clearly has limits. Lacking recursive features, it essentially must analyze long, rare constructions using only contexts. However, despite, or perhaps due to its simplicity, our model predicts bracketings very well, producing higher quality structural analyses than previous methods which employ the PCFG model family.

**Acknowledgements.**   We thank John Lafferty, Fernando Pereira, Ben Taskar, and Sebastian Thrun for comments and discussion. This paper is based on work supported in part by the National Science Foundation under Grant No. IIS-0085896.

## Footnotes

[1] We duplicated one of their experiments, which used grammars restricted to rules of the form $x \rightarrow x\ y \mid y\ x$, where there is one category $x$ for each part-of-speech (such a restricted CFG is isomorphic to a dependency grammar). We began reestimation from a grammar with uniform rewrite

[2]So, for the tree in figure 1(a), $P(t|s) \propto \exp(\lambda_{\text{NN NNS}} + \lambda_{\text{VBD IN NN}} + \lambda_{\text{IN NN}} + \lambda_{\diamond-\text{VBD}} + \lambda_{\text{NNS}-\diamond} + \lambda_{\text{VBD}-\diamond} + \lambda_{\diamond-\text{NNS}} + \lambda_{\text{NN}-\text{VBD}} + \lambda_{\text{NNS}-\text{IN}} + \lambda_{\text{VBD}-\text{NN}} + \lambda_{\text{IN}-\diamond})$.

[3]In practice, we stopped the system after 10 iterations, but final behavior was apparent after 4–8.

[4]In a sentence of length $n$, there are $(n+1)(n+2)/2$ total (possibly size zero) spans, but only $3n$ constituent spans: $n-1$ of size $\geq 2$, $n$ of size 1, and $n+1$ empty spans.

[5]Gaussian priors for the exponential model accomplish the former goal, but not the latter.

[6]The relative frequency estimators had a somewhat subtle positive effect. Empty spans have no effect on the model when using CG fitting, as all trees include the same empty spans. However, including their counts improved performance substantially when using relative frequency estimators. This is perhaps an indication that a generative version of this model would be advantageous.

[7]We discarded both brackets of length one and brackets spanning the entire sentence, since all of these are impossible to get incorrect, and hence ignored sentences of length $\leq 2$ during testing.

[8]For example, transitive sentences are bracketed [subject [verb object]] (*The president* [*executed the law*]) while nominalizations are bracketed [[possessive noun] complement] ([*The president's execution*] *of the law*), an arbitrary inconsistency which is unlikely to be learned automatically.

## References

[1] James K. Baker. Trainable grammars for speech recognition. In D. H. Klatt and J. J. Wolf, editors, *Speech Communication Papers for the 97th Meeting of the ASA*, pages 547–550, 1979.

[2] Glenn Carroll and Eugene Charniak. Two experiments on learning probabilistic dependency grammars from corpora. In C. Weir, S. Abney, R. Grishman, and R. Weischedel, editors, *Working Notes of the Workshop Statistically-Based NLP Techniques*, pages 1–13. AAAI Press, 1992.

[3] Eugene Charniak. A maximum-entropy-inspired parser. In *NAACL 1*, pages 132–139, 2000.

[4] Noam Chomsky. *Knowledge of Language*. Prager, New York, 1986.

[5] Noam Chomsky & Morris Halle. *The Sound Pattern of English*. Harper & Row, NY, 1968.

[6] Alexander Clark. Unsupervised induction of stochastic context-free grammars using distributional clustering. In *The Fifth Conference on Natural Language Learning*, 2001.

[7] Michael John Collins. Three generative, lexicalised models for statistical parsing. In *ACL 35/EACL 8*, pages 16–23, 1997.

[8] A.P. Dempster, N.M. Laird, and D.B. Rubin. Maximum likelihood from incomplete data via the EM algorithm. *J. Royal Statistical Society Series B*, 39:1–38, 1977.

[9] Steven Finch and Nick Chater. Distributional bootstrapping: From word class to proto-sentence. In *Proceedings of the Sixteenth Annual Conference of the Cognitive Science Society*, pages 301–306, Hillsdale, NJ, 1994. Lawrence Erlbaum.

[10] Zellig Harris. *Methods in Structural Linguistics*. University of Chicago Press, Chicago, 1951.

[11] Dan Klein and Christopher D. Manning. Distributional phrase structure induction. In *The Fifth Conference on Natural Language Learning*, 2001.

[12] K. Lari and S. J. Young. The estimation of stochastic context-free grammars using the inside-outside algorithm. *Computer Speech and Language*, 4:35–56, 1990.

[13] Fernando Pereira and Yves Schabes. Inside-outside reestimation from partially bracketed corpora. In *ACL 30*, pages 128–135, 1992.

[14] Andrew Radford. *Transformational Grammar*. Cambridge University Press, Cambridge, 1988.

[15] Hinrich Schütze. Distributional part-of-speech tagging. In *EACL 7*, pages 141–148, 1995.

[16] Andreas Stolcke and Stephen M. Omohundro. Inducing probabilistic grammars by Bayesian model merging. In *Grammatical Inference and Applications: Proceedings of the Second International Colloquium on Grammatical Inference*. Springer Verlag, 1994.

[17] M. van Zaanen and P. Adriaans. Comparing two unsupervised grammar induction systems: Alignment-based learning vs. emile. Technical Report 2001.05, University of Leeds, 2001.

[18] J. G. Wolff. Learning syntax and meanings through optimization and distributional analysis. In Y. Levy, I. M. Schlesinger, and M. D. S. Braine, editors, *Categories and processes in language acquisition*, pages 179–215. Lawrence Erlbaum, Hillsdale, NJ, 1988.